# Population Decoding Based on
# an Unfaithful Model

**S. Wu, H. Nakahara, N. Murata and S. Amari**
RIKEN Brain Science Institute
Hirosawa 2-1, Wako-shi, Saitama, Japan
{phwusi, hiro, mura, amari}@brain.riken.go.jp

## Abstract

We study a population decoding paradigm in which the maximum likelihood inference is based on an unfaithful decoding model (UMLI). This is usually the case for neural population decoding because the encoding process of the brain is not exactly known, or because a simplified decoding model is preferred for saving computational cost. We consider an unfaithful decoding model which neglects the pair-wise correlation between neuronal activities, and prove that UMLI is asymptotically efficient when the neuronal correlation is uniform or of limited-range. The performance of UMLI is compared with that of the maximum likelihood inference based on a faithful model and that of the center of mass decoding method. It turns out that UMLI has advantages of decreasing the computational complexity remarkably and maintaining a high-level decoding accuracy at the same time. The effect of correlation on the decoding accuracy is also discussed.

## 1 Introduction

Population coding is a method to encode and decode stimuli in a distributed way by using the joint activities of a number of neurons (e.g. Georgopoulos et al., 1986; Paradiso, 1988; Seung and Sompolinsky, 1993). Recently, there has been an expanded interest in understanding the population decoding methods, which particularly include the maximum likelihood inference (MLI), the center of mass (COM), the complex estimator (CE) and the optimal linear estimator (OLE) [see (Pouget et al., 1998; Salinas and Abbott, 1994) and the references therein]. Among them, MLI has an advantage of having small decoding error (asymptotic efficiency), but may suffers from the expense of computational complexity.

Let us consider a population of $N$ neurons coding a variable $x$. The encoding process of the population code is described by a conditional probability $q(\mathbf{r}|x)$ (Anderson, 1994; Zemel et al., 1998), where the components of the vector $\mathbf{r} = \{r_i\}$ for $i = 1, \cdots, N$ are the firing rates of neurons. We study the following MLI estimator given by the value of $x$ that maximizes the log likelihood $\ln p(\mathbf{r}|x)$, where $p(\mathbf{r}|x)$ is the decoding model which might be different from the encoding model $q(\mathbf{r}|x)$. So far, when people study MLI in a population code, it normally (*or implicitly*) assumes that $p(\mathbf{r}|x)$ is equal to the encoding model $q(\mathbf{r}|x)$. This requires that the estimator has full knowledge of the encoding process. Taking account of the complexity of the information process in the brain, it is more natural

to assume $p(\mathbf{r}|x) \neq q(\mathbf{r}|x)$. Another reason for choosing this is for saving computational cost. Therefore, a decoding paradigm in which the assumed decoding model is different from the encoding one needs to be studied. In the context of statistical theory, this is called estimation based on an unfaithful or a misspecified model. Hereafter, we call the decoding paradigm of using MLI based on an unfaithful model, UMLI, to distinguish from that of MLI based on the faithful model, which is called FMLI. The unfaithful model studied in this paper is the one which neglects the pair-wise correlation between neural activities. It turns out that UMLI has attracting properties of decreasing the computational cost of FMLI remarkably and at the same time maintaining a high-level decoding accuracy.

## 2 The Population Decoding Paradigm of UMLI

### 2.1 An Unfaithful Decoding Model of Neglecting the Neuronal Correlation

Let us consider a pair-wise correlated neural response model in which the neuron activities are assumed to be multivariate Gaussian

$$q(\mathbf{r}|x) = \frac{1}{\sqrt{(2\pi\sigma^2)^N \det(\mathbf{A})}} \exp[-\frac{1}{2\sigma^2} \sum_{i,j} A_{ij}^{-1}(r_i - f_i(x))(r_j - f_j(x))], \quad (1)$$

where $f_i(x)$ is the tuning function. In the present study, we will only consider the radial symmetry tuning function.

Two different correlation structures are considered. One is the uniform correlation model (Johnson, 1980; Abbott and Dayan, 1999), with the covariance matrix

$$A_{ij} = \delta_{ij} + c(1 - \delta_{ij}), \quad (2)$$

where the parameter $c$ (with $-1 < c < 1$) determines the strength of correlation.

The other correlation structure is of limited-range (Johnson, 1980; Snippe and Koenderink, 1992; Abbott and Dayan, 1999), with the covariance matrix

$$A_{ij} = b^{|i-j|}, \quad (3)$$

where the parameter $b$ (with $0 < b < 1$) determines the range of correlation. This structure has translational invariance in the sense that $A_{ij} = A_{kl}$, if $|i - j| = |k - l|$.

The unfaithful decoding model, treated in the present study, is the one which neglects the correlation in the encoding process but keeps the tuning functions unchanged, that is,

$$p(\mathbf{r}|x) = \frac{1}{\sqrt{(2\pi\sigma^2)^N}} \exp[-\frac{1}{2\sigma^2} \sum_{i} (r_i - f_i(x))^2]. \quad (4)$$

### 2.2 The decoding error of UMLI and FMLI

The decoding error of UMLI has been studied in the statistical theory (Akahira and Takeuchi, 1981; Murata et al., 1994). Here we generalize it to the population coding. For convenience, some notations are introduced. $\nabla f(\mathbf{r}, x)$ denotes $df(\mathbf{r}, x)/dx$. $E_q[f(\mathbf{r}, x)]$ and $V_q[f(\mathbf{r}, x)]$ denote, respectively, the mean value and the variance of $f(\mathbf{r}, x)$ with respect to the distribution $q(\mathbf{r}|x)$. Given an observation of the population activity $\mathbf{r}^\star$, the UMLI estimate $\hat{x}$ is the value of $x$ that maximizes the log likelihood $L_p(\mathbf{r}^\star, x) = \ln p(\mathbf{r}^\star|x)$.

Denote by $x_{\text{opt}}$ the value of $x$ satisfying $E_q[\nabla L_p(\mathbf{r}, x_{\text{opt}})] = 0$. For the faithful model where $p = q$, $x_{\text{opt}} = x$. Hence, $(x_{\text{opt}} - x)$ is the error due to the unfaithful setting, whereas $(\hat{x} - x_{\text{opt}})$ is the error due to sampling fluctuations. For the unfaithful model (4),

since $E_q[\nabla L_p(\mathbf{r}, x_{\mathrm{opt}})] = 0$, $\sum_i [f_i(x) - f_i(x_{\mathrm{opt}})] f_i'(x_{\mathrm{opt}}) = 0$. Hence, $x_{\mathrm{opt}} = x$ and UMLI gives an unbiased estimator in the present cases.

Let us consider the expansion of $\nabla L_p(\mathbf{r}^\star, \hat{x})$ at $x$,

$$\nabla L_p(\mathbf{r}^\star, \hat{x}) \simeq \nabla L_p(\mathbf{r}^\star, x) + \nabla \nabla L_p(\mathbf{r}^\star, x) \, (\hat{x} - x). \tag{5}$$

Since $\nabla L_p(\mathbf{r}^\star, \hat{x}) = 0$,

$$\frac{1}{N} \nabla \nabla L_p(\mathbf{r}^\star, x) \, (\hat{x} - x) \simeq -\frac{1}{N} \nabla L_p(\mathbf{r}^\star, x), \tag{6}$$

where $N$ is the number of neurons. Only the large $N$ limit is considered in the present study.

Let us analyze the properties of the two random variables $\frac{1}{N} \nabla \nabla L_p(\mathbf{r}^\star, x)$ and $\frac{1}{N} \nabla L_p(\mathbf{r}^\star, x)$. We consider first the uniform correlation model.

For the uniform correlation structure, we can write

$$r_i^\star = f_i(x) + \sigma(\epsilon_i + \eta), \tag{7}$$

where $\eta$ and $\{\epsilon_i\}$, for $i = 1, \cdots, N$, are independent random variables having zero mean and variance $c$ and $1 - c$, respectively. $\eta$ is the common noise for all neurons, representing the uniform character of the correlation.

By using the expression (7), we get

$$\frac{1}{N} \nabla L_p(\mathbf{r}^\star, x) = \frac{1}{N\sigma} \sum_i \epsilon_i f_i'(x) + \frac{\eta}{N\sigma} \sum_i f_i'(x), \tag{8}$$

$$\frac{1}{N} \nabla \nabla L_p(\mathbf{r}^\star, x) = \frac{1}{N\sigma} \sum_i \epsilon_i f_i''(x) - \frac{1}{N\sigma^2} \sum_i f_i'(x)^2$$
$$+ \frac{\eta}{N\sigma} \sum_i f_i''(x). \tag{9}$$

Without loss of generality, we assume that the distribution of the preferred stimuli is uniform. For the radial symmetry tuning functions, $\frac{1}{N} \sum_i f_i'(x)$ and $\frac{1}{N} \sum_i f_i''(x)$ approaches zero when $N$ is large. Therefore, the correlation contributions (the terms of $\eta$) in the above two equations can be neglected. UMLI performs in this case as if the neuronal signals are uncorrelated.

Thus, by the weak law of large numbers,

$$\frac{1}{N} \nabla \nabla L_p(\mathbf{r}^\star, x) \simeq -\frac{1}{N\sigma^2} \sum_i f_i'(x)^2$$
$$= \frac{Q_p}{N}, \tag{10}$$

where $Q_p \equiv E_q[\nabla \nabla L_p(\mathbf{r}, x)]$.

According to the central limit theorem, $\nabla L_p(\mathbf{r}^\star, x)/N$ converges to a Gaussian distribution

$$\frac{1}{N} \nabla L_p(\mathbf{r}^\star, x) \sim N(0, \frac{1-c}{N^2\sigma^2} \sum_i f_i'(x)^2)$$
$$= N(0, \frac{G_p}{N^2}), \tag{11}$$

where $N(0, t^2)$ denoting the Gaussian distribution having zero mean and variance $t$, and $G_p \equiv V_q[\nabla L_p(\mathbf{r}, x)]$.

Combining the results of eqs.(6), (10) and (11), we obtain the decoding error of UMLI,

$$
\begin{aligned}
(\hat{x} - x)_{\text{UMLI}} & \sim N(0, Q_p^{-2} G_p), \\
& = N(0, \frac{(1-c)\sigma^2}{\sum_i f_i'(x)^2}).
\end{aligned} \tag{12}
$$

In the similar way, the decoding error of FMLI is obtained,

$$
\begin{aligned}
(\hat{x} - x)_{\text{FMLI}} & \sim N(0, Q_q^{-2} G_q), \\
& = N(0, \frac{(1-c)\sigma^2}{\sum_i f_i'(x)^2}),
\end{aligned} \tag{13}
$$

which has the same form as that of UMLI except that $Q_q$ and $G_q$ are now defined with respect to the faithful decoding model, i.e., $p(\mathbf{r}|x) = q(\mathbf{r}|x)$. To get eq.(13), the condition $\sum_i f_i'(x) = 0$ is used. Interestingly, UMLI and FMLI have the same decoding error. This is because the uniform correlation effect is actually neglected in both UMLI and FMLI.

Note that in FMLI, $Q_q = G_q = V_q[\nabla L_q(\mathbf{r}|x)]$ is the Fisher information. $Q_q^{-2} G_q$ is the Cramér-Rao bound, which is the optimal accuracy for an unbiased estimator to achieve. Eq.(13) shows that FMLI is asymptotically efficient. For an unfaithful decoding model, $Q_p$ and $G_p$ are usually different from the Fisher information. We call $Q_p^{-2} G_p$ the generalized Cramér-Rao bound, and UMLI quasi-asymptotically efficient if its decoding error approaches $Q_p^{-2} G_p$ asymptotically. Eq.(12) shows that UMLI is quasi-asymptotic efficient.

In the above, we have proved the asymptotic efficiency of FMLI and UMLI when the neuronal correlation is uniform. The result relies on the radial symmetry of the tuning function and the uniform character of the correlation, which make it possible to cancel the correlation contributions from different neurons. For general tuning functions and correlation structures, the asymptotic efficiency of UMLI and FMLI may not hold. This is because the law of large numbers (eq.(10)) and the central limit theorem (eq.(11)) are not in general applicable.

We note that for the limited-range correlation model, since the correlation is translational invariant and its strength decreases quickly with the dissimilarity in the neurons' preferred stimuli, the correlation effect in the decoding of FMLI and UMLI becomes negligible when $N$ is large. This ensures that the law of large numbers and the central limit theorem hold in the large $N$ limit. Therefore, UMLI and FMLI are asymptotically efficient. This is confirmed in the simulation in Sec.3.

When UMLI and FMLI are asymptotic efficient, their decoding errors in the large $N$ limit can be calculated according to the Cramér-Rao bound and the generalized Cramér-Rao bound, respectively, which are

$$
\langle (\hat{x} - x)^2 \rangle_{\text{UMLI}} \sim \frac{\sigma^2 \sum_{ij} A_{ij} f_i'(x) f_j'(x)}{[\sum_i (f_i'(x))^2]^2}, \tag{14}
$$

$$
\langle (\hat{x} - x)^2 \rangle_{\text{FMLI}} \sim \frac{\sigma^2}{\sum_{ij} A_{ij}^{-1} f_i'(x) f_j'(x)}. \tag{15}
$$

## 3   Performance Comparison

The performance of UMLI is compared with that of FMLI and of the center of mass decoding method (COM). The neural population model we consider is a regular array of $N$ neurons (Baldi and Heiligenberg, 1988; Snippe, 1996) with the preferred stimuli uniformly distributed in the range $[-D, D]$, that is, $c_i = -D + 2iD/(N+1)$, for $i = 1, \cdots, N$. The comparison is done at the stimulus $x = 0$.

COM is a simple decoding method without using any information of the encoding process, whose estimate is the averaged value of the neurons' preferred stimuli weighted by the responses (Georgopoulos et al., 1982; Snippe, 1996), i.e.,

$$\hat{x} = \frac{\sum_i r_i c_i}{\sum_i r_i}. \tag{16}$$

The shortcoming of COM is a large decoding error.

For the population model we consider, the decoding error of COM is calculated to be

$$\langle (\hat{x} - x)^2 \rangle_{\text{COM}} \sim \frac{\sigma^2 \sum_{ij} A_{ij} c_i c_j}{[\sum_i f_i(x)]^2}, \tag{17}$$

where the condition $\sum_i f_i(x) c_i = 0$ is used, due to the regularity of the distribution of the preferred stimuli.

The tuning function is Gaussian, which has the form

$$f_i(x) = \exp[-\frac{(x - c_i)^2}{2a^2}], \tag{18}$$

where the parameter $a$ is the tuning width.

We note that the Gaussian response model does not give zero probability for negative firing rates. To make it more reliable, we set $r_i = 0$ when $f_i(x) < 0.11$ ($|x - c_i| > 3a$), which means that only those neurons which are active enough contribute to the decoding. It is easy to see that this cut-off does not effect much the results of UMLI and FMLI, due to their nature of decoding by using the derivative of the tuning functions. Whereas, the decoding error of COM will be greatly enlarged without cut-off.

For the tuning width $a$, there are $N = \text{Int}[6a/d - 1]$ neurons involved in the decoding process, where $d$ is the difference in the preferred stimuli between two consecutive neurons and the function $\text{Int}[\cdot]$ denotes the integer part of the argument.

In all experiment settings, the parameters are chosen as $a = 1$ and $\sigma = 0.1$. The decoding errors of the three methods are compared for different values of $N$ when the correlation strength is fixed ($c = 0.5$ for the uniform correlation case and $b = 0.5$ for the limited-range correlation case), or different values of the correlation strength when $N$ is fixed to be 50.

Fig.1 compares the decoding errors of the three methods for the uniform correlation model. It shows that UMLI has the same decoding error as that of FMLI, and a lower error than that of COM. The uniform correlation improves the decoding accuracies of the three methods (Fig.1b).

In Fig.2, the simulation results for the decoding errors of FMLI and UMLI in the limited-range correlation model are compared with those obtained by using the Cramér-Rao bound and the generalized Cramér-Rao bound, respectively. It shows that the two results agree very well when the number of neurons, $N$, is large, which means that FMLI and UMLI are asymptotic efficient as we analyzed. In the simulation, the standard gradient descent method is used to maximize the log likelihood, and the initial guess for the stimulus is chosen as the preferred stimulus of the most active neuron. The CPU time of UMLI is around 1/5 of that of FMLI. UMLI reduces the computational cost of FMLI significantly.

Fig.3 compares the decoding errors of the three methods for the limited-range correlation model. It shows that UMLI has a lower decoding error than that of COM. Interestingly, UMLI has a comparable performance with that of FMLI for the whole range of correlation. The limited-range correlation degrades the decoding accuracies of the three methods when the strength is small and improves the accuracies when the strength is large (Fig.3b).

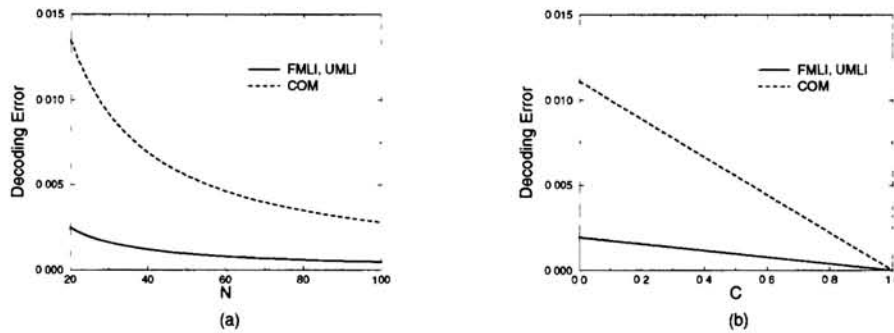

Figure 1: Comparing the decoding errors of UMLI, FMLI and COM for the uniform correlation model.

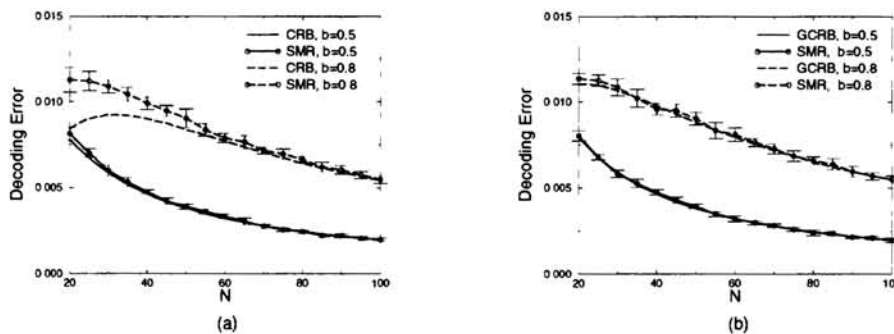

Figure 2: Comparing the simulation results of the decoding errors of UMLI and FMLI in the limited-range correlation model with those obtained by using the Cramér-Rao bound and the generalized Cramér-Rao bound, respectively. CRB denotes the Cramér-Rao bound, GCRB the generalized Cramér-Rao bound, and SMR the simulation result. In the simulation, 10 sets of data is generated, each of which is averaged over 1000 trials. (a) FMLI; (b) UMLI.

## 4   Discussions and Conclusions

We have studied a population decoding paradigm in which MLI is based on an unfaithful model. This is motivated by the facts that the encoding process of the brain is not exactly known by the estimator. As an example, we consider an unfaithful decoding model which neglects the pair-wise correlation between neuronal activities. Two different correlation structures are considered, namely, the uniform and the limited-range correlations. The performance of UMLI is compared with that of FMLI and COM. It turns out that UMLI has a lower decoding error than that of COM. Compared with FMLI, UMLI has comparable performance whereas with much less computational cost. It is our future work to understand the biological implication of UMLI.

As a by-product of the calculation, we also illustrate the effect of correlation on the decoding accuracies. It turns out that the correlation, depending on its form, can either improve or degrade the decoding accuracy. This observation agrees with the analysis of Abbott and Dayan (Abbott and Dayan, 1999), which is done with respect to the optimal decoding accuracy, i.e., the Cramér-Rao bound.

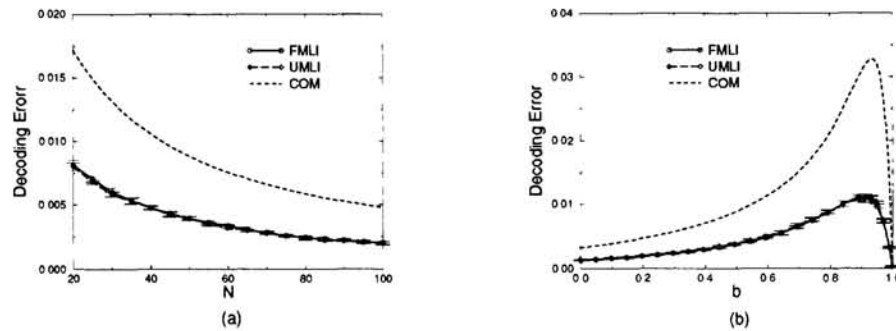

Figure 3: Comparing the decoding errors of UMLI, FMLI and COM for the limited-range correlation model.

## Acknowledgment

We thank the three anonymous reviewers for their valuable comments and insight suggestion. S. Wu acknowledges helpful discussions with Danmei Chen.

## References

L. F. Abbott and P. Dayan. 1999. The effect of correlated variability on the accuracy of a population code. *Neural Computation*, 11:91–101.

M. Akahira and K. Takeuchi. 1981. Asymptotic efficiency of statistical estimators: concepts and high order asymptotic efficiency. In *Lecture Notes in Statistics 7*.

C. H. Anderson. 1994. Basic elements of biological computational systems. *International Journal of Modern Physics C*, 5:135–137.

P. Baldi and W. Heiligenberg. 1988. How sensory maps could enhance resolution through ordered arrangements of broadly tuned receivers. *Biol. Cybern.*, 59:313–318.

A. P. Georgopoulos, J. F. Kalaska, R. Caminiti, and J. T. Massey. 1982. On the relations between the direction of two-dimensional arm movements and cell discharge in primate motor cortex. *J. Neurosci.*, 2:1527–1537.

K. O. Johnson. 1980. Sensory discrimination: neural processes preceding discrimination decision. *J. Neurophy.*, 43:1793–1815.

M. Murata, S. Yoshizawa, and S. Amari. 1994. Network information criterion-determining the number of hidden units for an artificial neural network model. *IEEE. Trans. Neural Networks*, 5:865–872.

A. Pouget, K. Zhang, S. Deneve, and P. E. Latham. 1998. Statistically efficient estimation using population coding. *Neural Computation*, 10:373–401.

E. Salinas and L. F. Abbott. 1994. Vector reconstruction from firing rates. *Journal of Computational Neuroscience*, 1:89–107.

H. P. Snippe and J. J. Koenderink. 1992. Information in channel-coded systems: correlated receivers. *Biological Cybernetics*, 67:183–190.

H. P. Snippe. 1996. Parameter extraction from population codes: a critical assessment. *Neural Computation*, 8:511–529.

R. S. Zemel, P. Dayan, and A. Pouget. 1998. Population interpolation of population codes. *Neural Computation*, 10:403–430.